# The Early Word Catches the Weights

Mark A. Smith          Garrison W. Cottrell          Karen L. Anderson
Department of Computer Science
University of California at San Diego
La Jolla, CA 92093
{*masmith,gary,kanders*}*@cs.ucsd.edu*

## Abstract

The strong correlation between the frequency of words and their naming latency has been well documented. However, as early as 1973, the Age of Acquisition (AoA) of a word was alleged to be the actual variable of interest, but these studies seem to have been ignored in most of the literature. Recently, there has been a resurgence of interest in AoA. While some studies have shown that frequency has no effect when AoA is controlled for, more recent studies have found independent contributions of frequency and AoA. Connectionist models have repeatedly shown strong effects of frequency, but little attention has been paid to whether they can also show AoA effects. Indeed, several researchers have explicitly claimed that they cannot show AoA effects. In this work, we explore these claims using a simple feed forward neural network. We find a significant contribution of AoA to naming latency, as well as conditions under which frequency provides an independent contribution.

## 1   Background

Naming latency is the time between the presentation of a picture or written word and the beginning of the correct utterance of that word. It is undisputed that there are significant differences in the naming latency of many words, even when controlling word length, syllabic complexity, and other structural variants. The cause of differences in naming latency has been the subject of numerous studies. Earlier studies found that the frequency with which a word appears in spoken English is the best determinant of its naming latency (Oldfield & Wingfield, 1965). More recent psychological studies, however, show that the age at which a word is learned, or its Age of Acquisition (AoA), may be a better predictor of naming latency. Further, in many multiple regression analyses, frequency is not found to be significant when AoA is controlled for (Brown & Watson, 1987; Carroll & White, 1973; Morrison et al. 1992; Morrison & Ellis, 1995). These studies show that frequency and AoA are highly correlated (typically r = -.6) explaining the confound of older studies on frequency. However, still more recent studies question this finding and find that both AoA and frequency are significant and contribute independently to naming latency (Ellis & Morrison, 1998; Gerhand & Barry, 1998,1999).

Much like their psychological counterparts, connectionist networks also show very strong frequency effects. However, the ability of a connectionist network to show AoA effects has been doubted (Gerhand & Barry, 1998; Morrison & Ellis, 1995). Most of these claims are

based on the well known fact that connectionist networks exhibit "destructive interference" in which later presented stimuli, in order to be learned, force early learned inputs to become less well represented, effectively increasing their associated errors. However, these effects only occur when training ceases on the early patterns. Continued training on all the patterns mitigates the effects of interference from later patterns.

Recently, Ellis & Lambon-Ralph (in press) have shown that when pattern presentation is staged, with one set of patterns initially trained, and a second set added into the training set later, strong AoA effects are found. They show that this result is due to a loss of plasticity in the network units, which tend to get out of the linear range with more training. While this result is not surprising, it is a good model of the fact that some words may not come into existence until late in life, such as "email" for baby boomers. However, they explicitly claim that it is important to stage the learning in this way, and offer no explanation of what happens during early word acquisition, when the surrounding vocabulary is relatively constant, or why and when frequency and AoA show independent effects.

In this paper, we present an abstract feed-forward computational model of word acquisition that does not stage inputs. We use this model to examine the effects of frequency and AoA on sum squared error, the usual variable used to model reaction time. We find a consistent contribution of AoA to naming latency, as well as the conditions under which there is an independent contribution from frequency in some tasks.

## 2    Experiment 1: Do networks show AoA effects?

Our first goal was to show that AoA effects could be observed in a connectionist network using the simplest possible model. First, we need to define AoA in a network. We did this is such a way that staging the inputs was not necessary: we defined a threshold for the error, after which we would say a pattern has been "acquired." The AoA is defined to be the epoch during which this threshold is crossed. Since error for a particular pattern may occasionally go up again during online learning, we also measured the *last* epoch that the pattern went below the threshold for final time. We analyzed our networks using both definitions of acquisition (which we call first acquisition and final acquisition), and have found that the results vary little between these definitions. In what follows, we use first acquisition for simplicity.

### 2.1    The Model

The simplest possible model is an autoencoder network. Using a network architecture of 20-15-20, we trained the network to autoencode 200 patterns of random bits (each bit had a 50% probability of being on or off). We initialized weights randomly with a flat distribution of values between 0.1 and -0.1, used a learning rate of 0.001 and momentum of 0.9.

For this experiment, we chose the AoA threshold to be 2, indicating an average squared error of .1 per input bit, yielding outputs much closer to the correct output than any other. We calculated Euclidean distances between all outputs and patterns to verify that the input was mapped most closely to the correct output. Training on the entire corpus continued until 98% of all patterns fell below this threshold.

### 2.2    Results

After the network had learned the input corpus, we investigated the relationship between the epoch at which the input vector had been learned and the final sum squared error (equivalent, for us, to "adult" naming latency) for that input vector. These results are presented in Figure 1. The relationship between the age of acquisition of the input vector and its

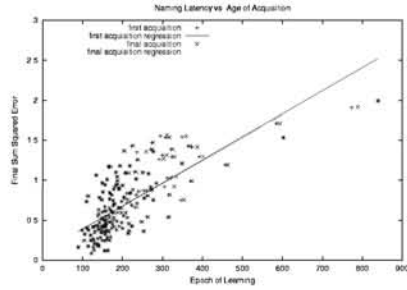

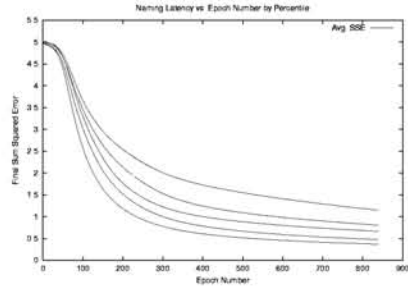

Figure 1: Exp. 1. Final SSE vs. AoA.      Figure 2: SSE vs. Epoch by Percentile

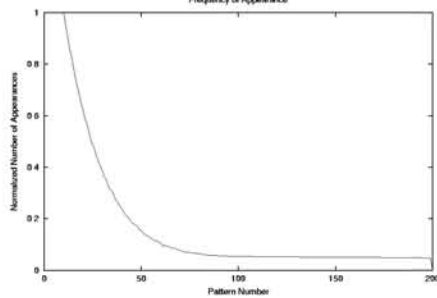

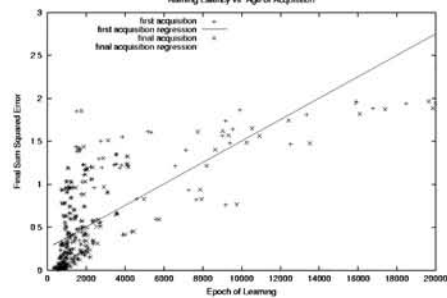

Figure 3: Exp. 2 Frequency Distribution      Figure 4: Exp. 2 Final SSE vs. AoA

final sum squared error is clear: the earlier an input is learned, the lower its final error will be. A more formal analysis of this relationship yields a significant ($p \ll .005$) correlation coefficient of r=0.749 averaged over 10 runs of the network.

In order to understand this relationship better, we divided the learned words into five percentile groups depending upon AoA. Figure 2 shows the average SSE for each group plotted over epoch number. The line with the least average SSE corresponds to the earliest acquired quintile while the line with the highest average SSE corresponds to the last acquired quintile. From this graph we can see that the average SSE for earlier learned patterns stays below errors for late learned patterns. This is true from the outset of learning as well as when the error starts to decrease less rapidly as it asymptotically approaches some lowest error limit. We sloganize this result as "the patterns that get to the weights first, win."

## 3   Experiment 2: Do AoA effects survive a frequency manipulation?

Having displayed that AoA effects are present in connectionist networks, we wanted to investigate the interaction with frequency. We model the frequency distribution of inputs after the known English spoken word frequency in which very few words appear very often while a very large portion of words appear very seldom (Zipf's law). The frequency distribution we used (presentation probability= $0.05 + 0.95 * ((1 - (1.0/numinputs) * input\_number) + 0.05)^{10}$ ) is presented in Figure 3 (a true version of Zipf's law still shows the result). Otherwise, all parameters are the same as Exp. 1.

### 3.1   Results

Results are plotted in Figure 4. Here we find again a very strong and significant ($p \ll 0.005$) correlation between the age at which an input is learned and its naming latency. The correlation coefficient averaged over 10 runs is 0.668. This fits very well with known data. Figure 5 shows how the frequency of presentation of a given stimulus correlates with

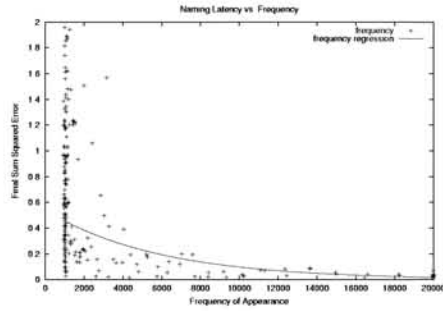
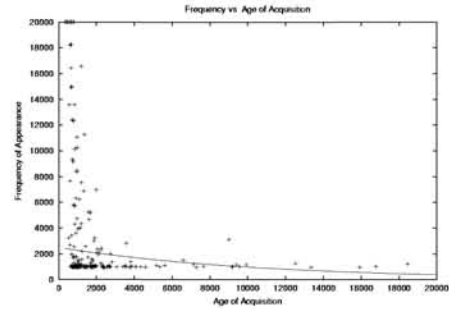

Figure 5: Exp. 2 Frequency vs. SSE          Figure 6: Exp. 2 AoA vs. Frequency

naming latency. We find that the best fitting correlation is an exponential one in which naming latency correlates most strongly with the log of the frequency. The correlation coefficient averaged over 10 runs is significant ($p \ll 0.005$) at -0.730. This is a slightly stronger correlation than is found in the literature.

Finally, figure 6 shows how frequency and AoA are related. Again, we find a significant ($p < 0.005$) correlation coefficient of -0.283 averaged over 10 runs. However, this is a much weaker correlation than is found in the literature. Performing a multiple regression with the dependent variable as SSE and the two explaining variables as AoA and log frequency, we find that both AoA and log frequency contribute significantly ($p \ll 0.005$ for both variables) to the regression equation. Whereas AoA correlates with SSE at 0.668 and log frequency correlates with SSE at -0.730, the multiple correlation coefficient averaged over 10 runs is 0.794. AoA and log frequency each make independent contributions to naming latency.

We were encouraged that we found both effects of frequency and AoA on SSE in our model, but were surprised by the small size of the correlation between the two. The naming literature shows a strong correlation between AoA and frequency. However, pilot work with a smaller network showed *no* frequency effect, which was due to the autoencoding task in a network where the patterns filled 20% of the input space (200 random patterns in a 10-8-10 network, with 1024 patterns possible). This suggests that autoencoding is not an appropriate task to model naming, and would give rise to the low correlation between AoA and frequency. Indeed, English spellings and their corresponding sounds are certainly correlated, but not completely consistent, with many exceptional mappings. Spelling-sound consistency has been shown to have a significant effect on naming latency (Jared, McRae, & Seidenberg, 1990). Object naming, another task in which AoA effects are found, is a completely arbitrary mapping. Our third experiment looks at the effect that consistency of our mapping task has on AoA and frequency effects.

## 4 Experiment 3: Consistency effects

Our model in this experiment is identical to the previous model except for two changes. First, to encode mappings with varying degrees of consistency, we needed to increase the number of hidden units to 50, resulting in a 20-50-20 architecture. Second, we found that some patterns would end up with one bit off, leading to a bimodal distribution of SSE's. We thus used cross-entropy error to ensure that all bits would be learned.

Eleven levels of consistency were defined; from 100% consistent, or autoencoding; to 0% consistent, or a mapping from one random 20 bit vector to another random 20 bit vector. Note that in a 0% consistent mapping, since each bit as a 50% chance of being on, about 50% of the bits will be the same by chance. Thus an intermediate level of 50% consistency will have on average 75% of the corresponding bits equal.

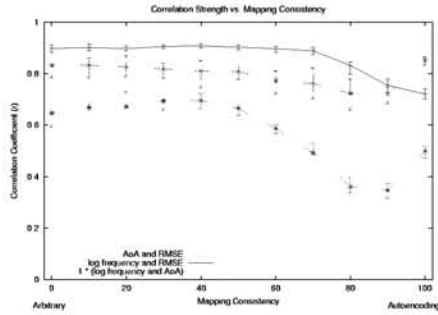
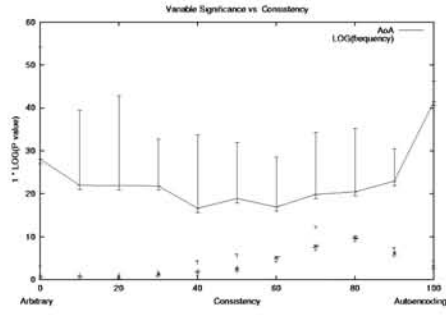

Figure 7: Exp. 3 R-values vs. Consistency   Figure 8: Exp. 3 P-values vs. Consistency

## 4.1 Results

Using this scheme, ten runs at each consistency level were performed. Correlation coefficients between AoA and naming latency (RMSE), log(frequency) and naming latency, and AoA and log(frequency) were examined. These results can be found in Figure 7. It is clear that AoA exhibits a strong effect on RMSE at all levels of consistency, peaking at a fully consistent mapping. We believe that this may be due to the weaker effect of frequency when all patterns are consistent, and each pattern is supporting the same mapping. Frequency also shows a strong effect on RMSE at all levels of consistency, with its influence being lowest in the autoencoding task, as expected. Most interesting is the correlation strength between AoA and frequency across consistency levels. While we do not yet have a good explanation for the dip in correlation at the 80-90% level of consistency, it provides a possible explanation of the multiple regression data we describe next.

Multiple regressions with the dependent variable as error and explaining variables as log(frequency) and AoA were performed. In Figure 8, we plot the negative log of the p-value of AoA and log(frequency) in the regression equation over consistency levels. Most notable is the finding that AoA is significant at extreme levels at all levels of consistency. A value of 30 on this plot corresponds to a p-value of $10^{-30}$. Significance of log frequency has a more complex interaction with consistency. Log frequency does not achieve significance in determining SSE until the patterns are almost 40% consistent. For more consistent mappings, however, significance increases dramatically to a P-value of less than $10^{-10}$ and then declines toward autoencoding. The data which may help us to explain what we see in Figure 8 actually lies in Figure 7. There is a relationship between log frequency significance and the correlation strength between AoA and log frequency. As AoA and frequency become less correlated, the significance of frequency increases, and vice-versa. Therefore, as frequency and AoA become less correlated, frequency is able to begin making an independent contribution to the SSE of the network. Such interactions may explain the sometimes inconsistent findings in the literature; depending upon the task and the individual items in the stimuli, different levels of consistency of mapping can affect the results. However, each of these points represent an average over a set of networks with one average consistency value. It is doubtful that any natural mapping, such as spelling to sound, has such a uniform distribution. We rectify this in the next experiment.

## 5   Experiment 4: Modelling spelling-sound correspondences

Our final experiment is an abstract simulation of learning to read, both in terms of word frequency and spelling-sound consistency. Most English words are considered consistent in their spelling-sound relationship. This depends on whether words in their spelling "neighborhood" agree with them in pronunciation, e.g., "cave," "rave," and "pave." However, a small but important portion of our vocabulary consists of inconsistent words, e.g., "have."

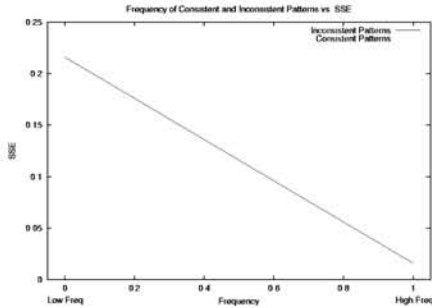
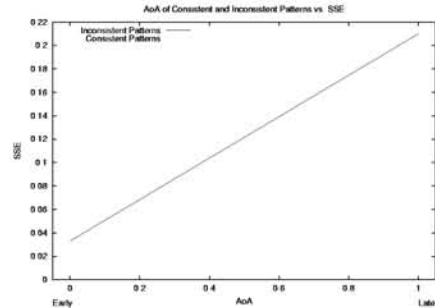

Figure 9: Exp. 4 Consistency vs. frequency     Figure 10: Exp. 4 Consistency AoA

The reason that "have" continues to be pronounced inconsistently is because it is a very frequent word. Inconsistent words have the property that they are on average much more frequent than consistent words although there are far more consistent words by number.

To model this we created an input corpus of 170 consistent words and 30 inconsistent words. Inconsistent words were arbitrarily defined as 50% consistent, or an average of 5 bit flips in a 20 bit pattern; consistent words were modeled as 80% consistent, or an average of 2 bit flips per pattern. The 30 inconsistent words were presented with high frequencies corresponding to the odd numbered patterns (1..59) in Figure 3. The even numbered patterns from 2 to 60 were the consistent words. The remaining patterns were also consistent. This allowed us to compare consistent and inconsistent words in the same frequency range, controlling for frequency in a much cleaner manner than is possible in human subject experiments. The network is identical to the one in Experiment 3.

## 5.1   Results

We first analyzed the data for the standard consistency by frequency interaction. We labeled the 15 highest frequency consistent and inconsistent patterns as "high frequency" and the next 15 of each as the "low frequency" patterns, in order to get the same number of patterns in each cell, by design. The results are shown in Figure 9, and show the standard interaction. More interestingly, we did a post-hoc median split of the patterns based on their AoA, defining them as "early" or "late" in this way, and then divided them based on consistency. This is shown in Figure 10. An ANOVA using unequal cell size corrections shows a significant ($p < .001$) interaction between AoA and consistency.

## 6   Discussion

Although the possibility of Age of Acquisition effects in connectionist networks has been doubted, we found a very strong, significant, and reproducible effect of AoA on SSE, the variable most often used to model reaction time, in our networks. Patterns which are learned in earlier epochs consistently show lower final error values than their late acquired counterparts. In this study, we have shown that this effect is present across various learning tasks, network topologies, and frequencies. Informally, we have found AoA effects across more network variants than reported here, including different learning rates, momentum, stopping criterion, and frequency distributions. In fact, across *all* runs we conducted for this study, we found strong AoA effects, provided the network was able to learn its task. We believe that this is because AoA is an intrinsic property of connectionist networks.

We have performed some preliminary analyses concerning which patterns are acquired early. Using the setup of Experiment 1, that is, autoencoded 20 bit patterns, we have found that the patterns that are most correlated with the other patterns in the training set tend to

be the earliest acquired, with $r^2 = 0.298$. (We should note that interpattern correlations are very small, but positive, because no bits are negative). Thus patterns that are most consistent with the training set are learned earliest. We have yet to investigate how this generalizes to arbitrary mappings, but, given the results of Experiment 4, it makes sense to predict that the most frequent, most consistently mapped patterns (e.g., in the largest spelling–sound neighborhood) would be the earliest acquired, in the absence of other factors.

## 7  Future Work

This study used a very general network and learning task to demonstrate AoA effects in connectionist networks. There is therefore no reason to suspect that this effect is limited to words, and indeed, AoA effects have been found in face recognition. Meanwhile, we have not investigated the interaction of our simple model of AoA effects with staged presentation. Presumably words acquired late are fewer in number, and Ellis & Lambon-Ralph (in press) have shown that they must be extremely frequent to overcome their lateness. Our results suggest that patterns that are most consistent with earlier acquired mappings would also overcome their lateness. We are particularly interested in applying these ideas to a realistic model English reading acquisition, where actual consistency effects can be measured in the context of friend/enemy ratios in a neighborhood. Finally, we would like to explore whether the AoA effect is universal in connectionist networks, or if under some circumstances AoA effects are not observed.

**Acknowledgements**

We would like to thank the Elizabeth Bates for alerting us to the work of Dr. Andrew Ellis, and for the latter for providing us with a copy of Ellis & Lambon-Ralph (in press).

**References**

[1] Brown, G.D.A. & Watson, F.L. (1987) First in, first out: Word learning age and spoken word frequency as predictors of word familiarity and naming latency. *Memory & Cognition, 15*:208-216

[2] Carroll, J.B. & White, M.N. (1973). Word frequency and age of acquisition as determiners of picture-naming latency. *Quarterly Journal of Psychology, 25* pp. 85-95

[3] Ellis, A.W. & Morrison, C.M. (1998). Real age of acquisition effects in lexical retrieval. *Journal of Experimental Psychology: Learning, Memory, & Cognition, 24* pp. 515-523

[4] Ellis, A.W. & Lambon Ralph, M.A. (in press). Age of Acquisition effects in adult lexical processing reflect loss of plasticity in maturing systems: Insights from connectionist networks. *JEP: LMC*.

[5] Gerhand, S. & Barry, C. (1998). Word frequency effects in oral reading are not merely Age-of-Acquisition effects in disguise. *JEP:LMC, 24* pp. 267-283.

[6] Gerhand, S. & Barry, C. (1999). Age of acquisition and frequency effects in speeded word naming. *Cognition, 73* pp. B27-B36

[7] Jared, D., McRae, K., & Seidenberg, M.S. (1990). The Basis of Consistency Effects in Word Naming. *JML, 29* pp. 687-715

[8] Morrison, C.M., Ellis, A.W. & Quinlan, P.T. (1992). Age of acquisition, not word frequency, affects object naming, nor object recognition. *Memory & Cognition, 20* pp. 705-714

[9] Morrison, C.M. & Ellis, A.W. (1995). Roles of Word Frequency and Age of Acquisition in Word Naming and Lexical Decision. *JEP:LMC, 21* pp. 116-133

[10] Oldfield, R.C. & Wingfield, A. (1965) Response latencies in naming objects. *Quarterly Journal of Experimental Psychology 17*, pp. 273-281.
